# Gaussian Processes for Multiuser Detection in CDMA receivers

**Juan José Murillo-Fuentes, Sebastian Caro**
Dept. Signal Theory and Communications
University of Seville
{murillo,scaro}@us.es

**Fernando Pérez-Cruz**
Gatsby Computational Neuroscience
University College London
fernando@gatsby.ucl.ac.uk

## Abstract

In this paper we propose a new receiver for digital communications. We focus on the application of Gaussian Processes (GPs) to the multiuser detection (MUD) in code division multiple access (CDMA) systems to solve the near-far problem. Hence, we aim to reduce the interference from other users sharing the same frequency band. While usual approaches minimize the mean square error (MMSE) to linearly retrieve the user of interest, we exploit the same criteria but in the design of a nonlinear MUD. Since the optimal solution is known to be nonlinear, the performance of this novel method clearly improves that of the MMSE detectors. Furthermore, the GP based MUD achieves excellent interference suppression even for short training sequences. We also include some experiments to illustrate that other nonlinear detectors such as those based on Support Vector Machines (SVMs) exhibit a worse performance.

## 1  Introduction

One of the major issues in present wireless communications is how users share the resources. And particularly, how they access to a common frequency band. Code division multiple access (CDMA) is one of the techniques exploited in third generation communications systems and is to be employed in the next generation. In CDMA each user uses direct sequence spread spectrum (DS-SS) to modulate its bits with an assigned code, spreading them over the entire frequency band. While typical receivers deal only with interferences and noise intrinsic to the channel (i.e. Inter-Symbolic Interference, intermodulation products, spurious frequencies, and thermal noise), in CDMA we also have interference produced by other users accessing the channel at the same time. Interference limitation due to the simultaneous access of multiple users systems has been the stimulus to the development of a powerful family of Signal Processing techniques, namely Multiuser Detection (MUD). These techniques have been extensively applied to CDMA systems. Thus, most of last generation digital communication systems such as Global Positioning System (GPS), wireless 802.11b, Universal Mobile Telecommunication System (UMTS), etc, may take advantage of any improvement on this topic.

In CDMA, we face the retrieval of a given user, the user of interest (UOI), with the knowledge of its associated code or even the whole set of users codes. Hence, we face the suppression of interference due to others users. If all users transmit with the same power,

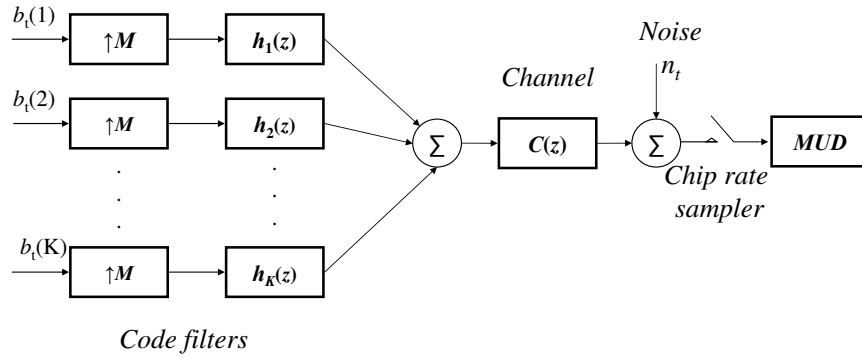

Figure 1: Synchronous CDMA system

but the UOI is far from the receiver, most users reach the receiver with a larger amplitude, making it more difficult to detect the bits of the UOI. This is well-known as the near-far problem. Simple detectors can be designed by minimizing the mean square error (MMSE) to linearly retrieve the user of interest [5]. However, these detectors need large sequences of training data. Besides, the optimal solution is known to be nonlinear.

There has been several attempts to solve the problem using nonlinear techniques. There are solutions based on Neural Networks such as multilayer perceptron or radial basis functions [1, 3], but training times are long and unpredictable. Recently, support vector machines (SVM) have been also applied to CDMA MUD [4]. This solution need very long training sequences (a few hundreds bits) and they are only tested in toy examples with very few users and short spreading sequences (the code for each user). In this paper, we will present a multiuser detector based on Gaussian Processes [7]. The MUD detector is inspired by the linear MMSE criteria, which can be interpreted as a Bayesian linear regressor. In this sense, we can extend the linear MMSE criteria to nonlinear decision functions using the same ideas developed in [6] to present Gaussian Processes for regression.

The rest of the paper is organised as follows. In Section 2, we present the multiuser detection problem in CDMA communication systems and the widely used minimum mean square error receiver. We propose a nonlinear receiver based on Gaussian Processes in Section 3. Section 4 is devoted to show, through computer experiments, the advantages of the GP-MUD receiver with short training sequences. We compare it to the linear MMSE and the nonlinear SVM MUD. We conclude the paper in Section 5 presenting some remarks and future work.

## 2    CDMA Communication System Model and MUD

Consider a synchronous CDMA digital communication system [5] as depicted in Figure 1. Its main goal is to share the channel between different users, discriminating between them by different assigned codes. Each transmitted bit is upsampled and multiplied by the users' spreading codes and then the chips for each bit are transmitted into the channel (each element of the spreading code is either $+1$ or $-1$ and they are known as chips). The channel is assumed to be linear and noisy, therefore the chips from different users are added together, plus Gaussian noise. Hence, the MUD has to recover from these chips the bits corresponding to each user. At each time step $t$, the signal in the receiver can be represented

in matrix notation as:

$$\boldsymbol{x}_t = \boldsymbol{H}\boldsymbol{A}\boldsymbol{b}_t + \boldsymbol{n}_t \qquad (1)$$

where $\boldsymbol{b}_t$ is a column vector that contains the bits ($+1$ or $-1$) for the $K$ users at time $k$. The $K \times K$ diagonal matrix $\boldsymbol{A}$ contains the amplitude of each user, which represents the attenuation that each user's transmission suffers through the channel (this attenuation depends on the distance between the user and the receiver). $\boldsymbol{H}$ is an $L \times K$ matrix which contains in each column the $L$-dimensional spreading code for each of the $K$ users. The spreading codes are designed to present a low cross-correlation between them and between any shifted version of the codes, to guarantee that the bits from each user can be readily recovered. The codes are known as spreading sequences, because they augment the occupied bandwidth of the transmitted signal by $L$. Finally, $\boldsymbol{x}_t$ represents the $L$ received chips to which Gaussian noise has been added, which is denoted by $\boldsymbol{n}_t$.

At reception, we aim to estimate the original transmitted symbols of any user $i$, $\boldsymbol{b}_t(i)$, hereafter the user of interest. Linear MUDs estimate these bits as

$$\hat{\boldsymbol{b}}_t(i) = sgn\{\boldsymbol{w}_i^\top \boldsymbol{x}_t\} \qquad (2)$$

The matched filter (MF) $\boldsymbol{w}_i = \boldsymbol{h}_i$, a simple correlation between $\boldsymbol{x}_t$ and the $i^{th}$ spreading code, is the optimal receiver if there were no additional users in the system, i.e. the received signal is only corrupted by Gaussian noise. The near-far problem arises when remaining users, apart from the UOI, are received with significantly higher amplitude. While the optimal solution is known to be nonlinear [5], some linear receivers such as the minimum mean square error (MMSE) present good performances and are used in practice. The MMSE receiver for the $i^{th}$ user solves:

$$\mathbf{w}_i^* = \arg\min_{\mathbf{w}_i} E\left[(\boldsymbol{b}_t(i) - \mathbf{w}_i^\top \boldsymbol{x}_t)^2\right] = \arg\min_{\mathbf{w}_i} E\left[(\boldsymbol{b}_t(i) - \mathbf{w}_i^\top (\boldsymbol{H}\boldsymbol{A}\boldsymbol{b}_t + \boldsymbol{\nu}_k))^2\right] \quad (3)$$

where $\mathbf{w}_i$ represents the decision function of the linear classifier. We can derive the MMSE receiver by taking derivatives with respect to $\mathbf{w}_i$ and equating to zero, obtaining:

$$\mathbf{w}_i^{MMSE_{de}} = \boldsymbol{R}_{xx}^{-1}\boldsymbol{h}_i \qquad (4)$$

where $R_{xx} = \mathrm{E}[\boldsymbol{x}_t\boldsymbol{x}_t^\top]$ is the correlation between the received vectors and $\boldsymbol{h}_i$ represents the spreading sequence of the UOI. This receiver is known as the decentralized MMSE receiver as it can be implemented without knowing the spreading sequences of the remaining users. Its main limitation is its performance, which is very low even for high signal to noise ratio, and it needs many examples (thousands) before it can recover the received symbols.

If the spreading codes of all the users are available, as in the base station, this information can be used to improve the performance of the MMSE detector. We can define $\boldsymbol{z}_k = \boldsymbol{H}^\top \boldsymbol{x}_t$, which is a vector of sufficient statistics for this problem [5]. The vector $\boldsymbol{z}_k$ is the matched-filter output for each user and it reduces the dimensionality of our problem from the number of chips $L$ to the number of users $K$, which is significantly lower in most applications. In this case the receiver is known as the centralized detector and it is defined as:

$$\mathbf{w}_i^{MMSE_{cent}} = \boldsymbol{H}\boldsymbol{R}_{zz}^{-1}\boldsymbol{H}^\top\boldsymbol{h}_i \qquad (5)$$

where $R_{zz} = \mathrm{E}[\boldsymbol{z}_t\boldsymbol{z}_t^\top]$ is the correlation matrix of the received chips after the MFs.

These MUDs have good convergence properties and do not need a training sequence to decode the received bits, but they need large training sequences before their probability of error is low. Therefore the initially received bits will present a very high probability of error that will make impossible to send any information on them. Some improvements can be achieved by using higher order statistics [2], but still the training sequences are not short enough for most applications.

## 3 Gaussian Processes for Multiuser Detection

The MMSE detector minimizes the functional in (3), which gives the best linear classifier. As we know, the optimal classifier is nonlinear [5], and the MMSE criteria can be readily extended to provide nonlinear models by mapping the received chips to a higher dimensional space. In this case we will need to solve:

$$\mathbf{w}_i^* = \arg \min_{\mathbf{w}_i} \left\{ \sum_{k=1}^{N} \left( \boldsymbol{b}_t(i) - \mathbf{w}_i^\top \boldsymbol{\phi}(\boldsymbol{x}_t) \right)^2 + \lambda \|\mathbf{w}_i\|^2 \right\} \tag{6}$$

in which we have changed the expectation by the empirical mean over a training set and we have incorporated a regularizer to avoid overfitting. $\boldsymbol{\phi}(\cdot)$ represents the nonlinear mapping of the received chips. The $\mathbf{w}_i$ that minimizes (6) can be interpreted as the mode of the parameters in a Bayesian linear regressor, as noted in [6], and since the likelihood and the prior are both Gaussians, so it will be the posterior. For any received symbol $\boldsymbol{x}_*$, we know that it will be distributed as a Gaussian with mean:

$$\mu(\boldsymbol{x}_*) = \frac{1}{\lambda} \boldsymbol{\phi}^\top(\boldsymbol{x}_*) \mathbf{A}^{-1} \boldsymbol{\Phi}^\top \mathbf{b} \tag{7}$$

and variance

$$\sigma^2(\boldsymbol{x}_*) = \boldsymbol{\phi}^\top(\boldsymbol{x}_*) \mathbf{A}^{-1} \boldsymbol{\phi}(\boldsymbol{x}_*) \tag{8}$$

where $\boldsymbol{\Phi} = [\boldsymbol{\phi}(\boldsymbol{x}_1), \boldsymbol{\phi}(\boldsymbol{x}_2), \dots, \boldsymbol{\phi}(\boldsymbol{x}_N)]^\top$, $\mathbf{b} = [\boldsymbol{b}_1(i), \boldsymbol{b}_2(i), \dots, \boldsymbol{b}_N(i)]^\top$ and $\mathbf{A} = \boldsymbol{\Phi}^\top \boldsymbol{\Phi} + \frac{1}{\lambda} \mathbf{I}$.

In the case the nonlinear mapping is unknown, we can still obtain the mean and variance for each received sample using the kernel of the transformation, being the mean:

$$\mu(\boldsymbol{x}_*) = \mathbf{k}^\top \mathbf{P}^{-1} \mathbf{b} \tag{9}$$

and variance

$$\sigma^2(\boldsymbol{x}_*) = k(\boldsymbol{x}_*, \boldsymbol{x}_*) + \mathbf{k}^\top \mathbf{P}^{-1} \mathbf{k} \tag{10}$$

where $k(\cdot, \cdot) = \boldsymbol{\phi}^\top(\cdot) \boldsymbol{\phi}(\cdot)$ is the kernel of the nonlinear transformation, $\mathbf{k} = [k(\boldsymbol{x}_*, \boldsymbol{x}_1), k(\boldsymbol{x}_*, \boldsymbol{x}_2), \dots, k(\boldsymbol{x}_*, \boldsymbol{x}_N)]$, and

$$\mathbf{P} = \boldsymbol{\Phi} \boldsymbol{\Phi}^\top + \lambda \mathbf{I} = \mathbf{K} + \lambda \mathbf{I} \tag{11}$$

where $(\mathbf{K})_{k\ell} = k(\boldsymbol{x}_t, \boldsymbol{x}_\ell)$. The kernel that we will use in our experiments are:

$$k(\boldsymbol{x}_t, \boldsymbol{x}_\ell) = e^{\theta[1]} \exp(-e^{\theta[4]} \|\boldsymbol{x}_t - \boldsymbol{x}_\ell\|^2) + e^{\theta[3]} \boldsymbol{x}_t^\top \boldsymbol{x}_\ell + e^{\theta[2]} \delta_{r,\ell} \tag{12}$$

The covariance function in (12) is a good kernel for solving the GP-MUD, because it contains a linear and a nonlinear part. The optimal decision surface for MUD is nonlinear, unless the spreading codes are orthogonal to each other, and its deviation from the linear solution depends on how strong the correlations between codes are. In most cases, a linear detector is very close to the optimal decision surface, as spreading codes are almost orthogonal, and only a minor correction is needed to achieve the optimal decision boundary. In this sense the proposed GP covariance function is ideal for the problem. The linear part can mimic the best linear decision boundary and the nonlinear part modifies it, where the linear explanation is not optimal. Also using a radial basis kernel for the nonlinear part is a good choice to achieve nonlinear decisions. Because, the received chips form a constellation of $2^K$ clouds of points with Gaussian spread around its centres.

Picturing the receiver as a Gaussian Process for regression, instead of a Regularised Least Square functional, allows us to either obtain the hyperparameters by maximizing the likelihood or marginalised them out using Monte Carlo techniques, as explained in [6]. For the

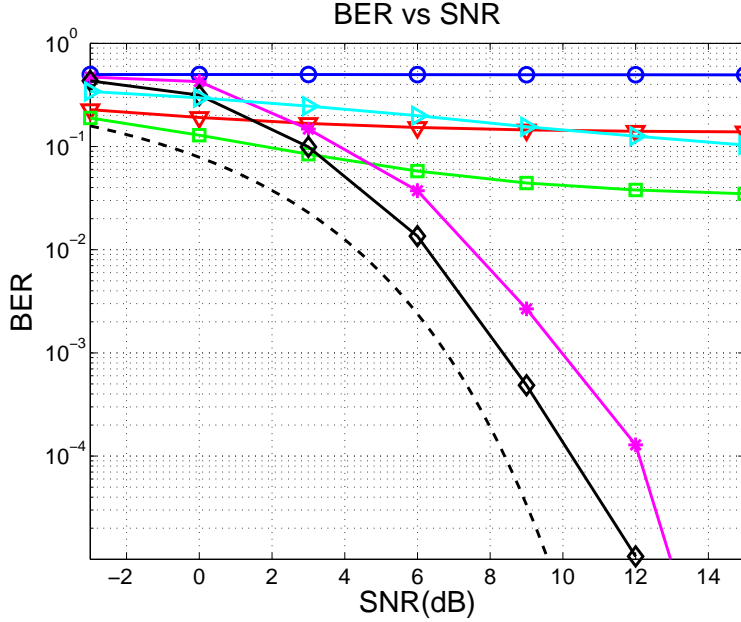

Figure 2: Bit Error Rate versus Signal to Noise ratio for the MF ($\triangledown$), MMSE-Centralized ($\square$), MMSE-Decentralized ($\circ$), SVM-centralized ($\triangleright$), GP-Centralized ($\diamond$) and GP-Decentralized ($*$) with $k = 8$ users and $n = 30$ training samples. The powers of the interfering users is distributed homogeneously between 0 and 30 dB above that of the UOI.

problem at hand speed is a must and we will be using the maximum likelihood hyperparameters.

We have just shown above how we can make predictions in the nonlinear case (9) using the received symbols from the channel. In an analogy with the MMSE receiver, this will correspond to the decentralized GP-MUD detector as we will not need to know the other users' codes to detect the bits sent to us. It is also relevant to notice that we do not need our spreading code for detection, as the decentralized MMSE detector did. We can also obtain a centralized GP-MUD detector using as input vectors $\boldsymbol{z}_t = \boldsymbol{H}^\top \boldsymbol{x}_t$.

## 4 Experiments

In this section we include the typical evaluation of the performance in a digital communications system, i.e., Bit Error Rate (BER). The test environment is a synchronous CDMA system in which the users are spread using Gold sequences with spreading factor $L = 31$ and $K = 8$ users, which are typical values in CDMA based mobile communication systems. We consider the same amplitude matrix in all experiments. These amplitudes are random values to achieve an interferer to signal ratio of 30 dB. Hence, the interferers are 30 dB over the UOI. We study the worse scenario and hence we will detect the user which arrives to the receiver with the lowest amplitude.

We compare the performance of the GP centralized and decentralized MUDs to the performance of the MMSE detectors, the Matched Filter detector and the (centralized) SVM-MUD in [4]. The SVM-MUD detector uses a Gaussian kernel and its width is adapted incorporating knowledge of the noise variance in the channel. We found that this setting

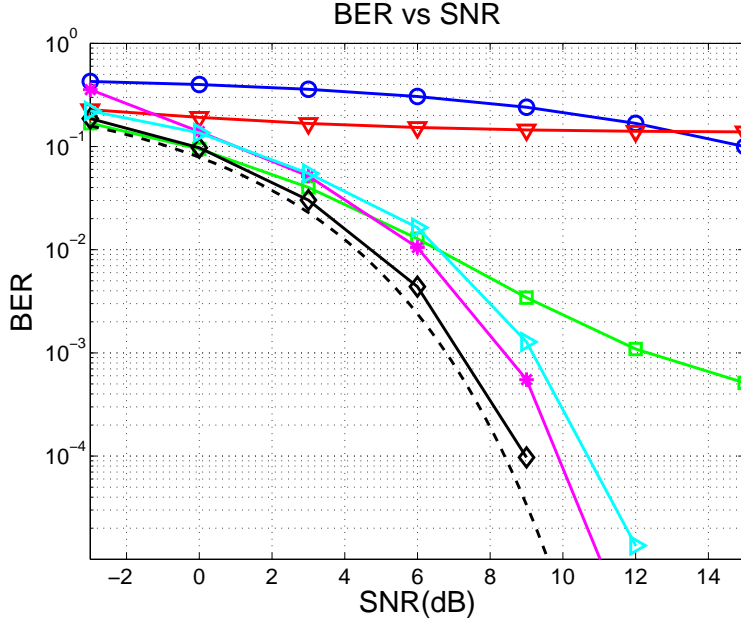

Figure 3: Bit Error Rate versus Signal to Noise ratio for the MF ($\triangledown$), MMSE-Centralized ($\square$), MMSE-Decentralized ($\circ$), SVM-centralized ($\triangleright$), GP-Centralized ($\diamond$) and GP-Decentralized ($*$) with $k = 8$ users and $n = 80$ training samples. The powers of the interfering users is distributed homogeneously between 0 and 30 dB above that of the UOI.

usually does not perform well for this experimental specification and we have set them using validation. We believe this might be due to either the reduced number of users in their experiments (2 or 3) or because they used the same amplitude for all the users, so they did not encounter the near-far problem.

We have included three experiments in which we have defined the number of training experiments equal to 30, 80 and 160. For each training set we have computed the BER for $10^6$ bits. The reported results are mean curves for 50 different trials.

The results in Figure 2 show that the detectors based on GPs are able to reduce the probability of error as the signal to noise ratio in the channel decreases with only 30 samples in the training sequence. The GP centralized MUD is only 1.5-2dB worse than the best achievable probability of error, which is obtained in absence of interference (indicated by the dashed line). The GP decentralized MUD reduces the probability of error as the signal to noise increases, but it remains between 3-4dB from the optimal performance. The other detectors are not able to decrease the BER even for a very high signal to noise ratio in the channel. These figures show that the GP based MUD can outperform the other MUD when very short training sequences are available.

Figure 3 highlights that the SVM-MUD (centralized) and the MSSE centralized detectors are able to reduce the BER as the SNR increases, but they are still far from the performance of the GP-MUD. The centralized GP-MUD basically provides optimal performance as it is less than 0.3db from the possible achieved BER when there is no interference in the channel. The decentralized GP-MUD outperforms the other two centralized detectors (SVM and MMSE) since it is able to provide lower BER without needing to know the code of the remaining users.

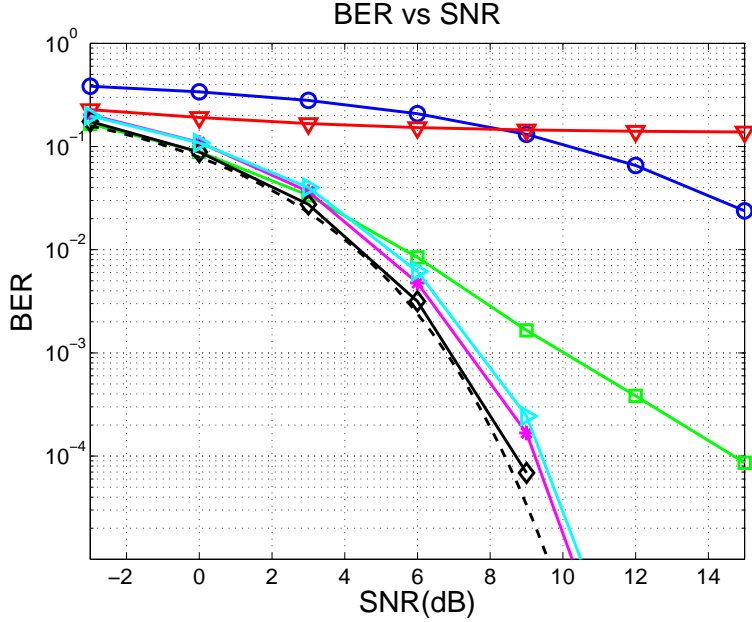

Figure 4: Bit Error Rate versus Signal to Noise ratio for the MF ($\triangledown$), MMSE-Centralized ($\square$), MMSE-Decentralized ($\circ$), SVM-centralized ($\triangleright$), GP-Centralized ($\diamond$) and GP-Decentralized ($*$) with $k = 8$ users and $n = 160$ training samples. The powers of the interfering users is distributed homogeneously between 0 and 30 dB above that of the UOI.

Finally, in Figure 4 we include the results for 160 training samples. In this case, the centralized GP-MUD lies above the optimal BER curve and the decentralized GP-MUD performs as the SVM-MUD detector. The centralized MMSE detector still presents very high probability of error for high signal to noise ratios and we need over 500 samples to obtain a performance similar to the centralized GP with 80 samples. For 160 samples the MMSE decentralized is already able to slightly reduce the bit error rate for very high signal to noise ratios. But to achieve the performance showed by the decentralized GP-MUD it needs several thousands samples.

## 5   Conclusions and Further Work

We propose a novel approach based on Gaussian Processes for regression to solve the near-far problem in CDMA receivers. Since the optimal solution is known to be nonlinear the Gaussian Processes are able to obtain this nonlinear decision surface with very few training examples. This is the main advantage of this method as it only requires a few tens training examples instead of the few hundreds needed by other nonlinear techniques as SVMs. This will allow its application in real communication systems, as training sequence of 26 samples are typically used in the GSM standard for mobile Telecommunications.

The most relevant result of this paper is the performance shown by the decentralized GP-MUD receiver, since it can be directly used over any CDMA system. The decentralized GP-MUD receiver does not need to know the codes from the other users and does not require the users to be aligned, as the other methods do. While the other receiver will degrade its performance if the users are not aligned, the decentralized GP-MUD receiver will not, providing a more robust solution to the near far problem.

We have presented some preliminary work, which shows that GPs for regression are suitable for the near-far problem in MUD. We have left for further work a more extensive set of experiments changing other parameters of the system such as: the number of users, the length of the spreading code, and the interferences with other users. But still, we believe the reported results are significant since we obtain low bit error rates for training sequences as short as 30 bits.

## Acknowledgements

Fernando Pérez-Cruz is Supported by the Spanish Ministry of Education Postdoctoral Fellowships EX2004-0698. This work has been partially funded by research grants TIC2003-02602 and TIC2003-03781 by the Spanish Ministry of Education.

## References

[1] G. C. Orsak B. Aazhang, B. P. Paris. Neural networks for multiuser detection in code-division multiple-access communications. *IEEE Transactions on Communications*, 40:1212–1222, 1992.

[2] Antonio Caamaño-Fernandez, Rafael Boloix-Tortosa, Javier Ramos, and Juan J. Murillo-Fuentes. High order statistics in multiuser detection. *IEEE Trans. on Man and Cybernetics C. Accepted for publication*, 2004.

[3] U. Mitra and H. V. Poor. Neural network techniques for adaptive multiuser demodulation. *IEEE Journal Selected Areas on Communications*, 12:14601470, 1994.

[4] L. Hanzo S. Chen, A. K. Samingan. Support vector machine multiuser receiver for DS-CDMA signals in multipath channels. *IEEE Transactions on Neural Network*, 12(3):604–611, December 2001.

[5] S. Verdú. *Multiuser Detection*. Cambridge University Press, 1998.

[6] C. Williams. Prediction with gaussian processes: From linear regression to linear prediction and beyond.

[7] Christopher K. I. Williams and Carl Edward Rasmussen. Gaussian processes for regression. In David S. Touretzky, Michael C. Mozer, and Michael E. Hasselmo, editors, *Proc. Conf. Advances in Neural Information Processing Systems, NIPS*, volume 8. MIT Press, 1995.
